# Predictive Indexing for Fast Search

**Sharad Goel**
Yahoo! Research
New York, NY 10018
goel@yahoo-inc.com

**John Langford**
Yahoo! Research
New York, NY 10018
jl@yahoo-inc.com

**Alex Strehl**
Yahoo! Research
New York, NY 10018
strehl@yahoo-inc.com

## Abstract

We tackle the computational problem of query-conditioned search. Given a machine-learned scoring rule and a query distribution, we build a predictive index by precomputing lists of potential results sorted based on an expected score of the result over future queries. The predictive index datastructure supports an anytime algorithm for approximate retrieval of the top elements. The general approach is applicable to webpage ranking, internet advertisement, and approximate nearest neighbor search. It is particularly effective in settings where standard techniques (e.g., inverted indices) are intractable. We experimentally find substantial improvement over existing methods for internet advertisement and approximate nearest neighbors.

## 1 Introduction

**The Problem.** The objective of *web search* is to quickly return the set of most relevant web pages given a particular query string. Accomplishing this task for a fixed query involves both determining the relevance of potential pages and then searching over the myriad set of all pages for the most relevant ones. Here we consider only the second problem. More formally, let $Q \subseteq \mathbb{R}^n$ be an *input space*, $W \subseteq \mathbb{R}^m$ a finite *output space* of size $N$, and $f : Q \times W \mapsto \mathbb{R}$ a known *scoring function*. Given an input (search query) $q \in Q$, the goal is to find, or closely approximate, the top-$k$ output objects (web pages) $p_1, \ldots, p_k$ in $W$ (i.e., the top $k$ objects as ranked by $f(q, \cdot)$).

The extreme speed constraint, often 100ms or less, and the large number of web pages ($N \approx 10^{10}$) makes web search a computationally-challenging problem. Even with perfect 1000-way parallelization on modern machines, there is far too little time to directly evaluate against every page when a particular query is submitted. This observation limits the applicability of machine-learning methods for building ranking functions. The question addressed here is: "Can we quickly return the highest scoring pages as ranked by complex scoring rules typical of learning algorithms?"

**Predictive Indexing.** We describe a method for rapidly retrieving the top elements over a large set as determined by general scoring functions. The standard method for mitigating the computational difficulties of search is to pre-process the data so that far less computation is necessary at runtime. Taking the empirical probability distribution of queries into account, we pre-compute collections of web pages that have a large expected score conditioned on the query falling into particular sets of related queries $\{Q_i\}$. For example, we may pre-compute and store the list of web pages that have the highest average score when the query contains the phrase "machine learning". To yield a practical algorithm, these sets should form meaningful groups of pages with respect to the scoring function and query distribution. At runtime, we then optimize only over those collections of top-scoring web pages for sets $Q_i$ containing the submitted query.

Our main contribution is optimizing the search index with respect to the query distribution. The empirical evidence presented shows that predictive indexing is an effective technique, making general machine learning style prediction methods viable for quickly ranking over large numbers of objects.

The general methodology applies to other optimization problems as well, including approximate nearest neighbor search.

In the remainder of Section 1 we describe existing solutions to large-scale search, and their applicability to general scoring functions. Section 2 describes the predictive indexing algorithm and covers an example and lemma suggesting that predictive indexing has significant advantages over existing techniques. We present empirical evaluation of the method in Section 3, using both proprietary web advertising data and public data for nearest neighbor search.

## 1.1 Feature Representation

One concrete way to map web search into the general predictive index framework is to represent both queries and pages as sparse binary feature vectors in a high-dimensional Euclidean space. Specifically, we associate each word with a coordinate: A query (page) has a value of $1$ for that coordinate if it contains the word, and a value of $0$ otherwise. We call this the *word-based feature representation*, because each query and page can be summarized by a list of its features (i.e., words) that it contains. The general predictive framework supports many other possible representations, including those that incorporate the difference between words in the title and words in the body of the web page, the number of times a word occurs, or the IP address of the user entering the query.

## 1.2 Related Work

Given the substantial importance of large-scale search, a variety of techniques have been developed to address the rapid ranking problem. Past work that has referenced the query distribution includes (Cheng et al., 2006; Chierichetti et al., 2008). Here we describe two commonly applied methods related to the predictive index approach.

**Fagin's Threshold Algorithm.** Fagin's threshold algorithm (Fagin et al., 2003) supports the top-$k$ problem for linear scoring functions of the form $f(q,p) = \sum_{i=1}^{n} q_i g_i(p)$, where $q_i \in \{0,1\}$ is the $i^{th}$ coordinate of the query $q$, and $g_i : W \mapsto \mathbb{R}$ are partial scores for pages as determined by the $i^{th}$ feature[1]. For each query feature $i$, construct an ordered list $L_i$ containing every web page, sorted in descending order by their partial scores $g_i(p)$. We refer to this as the *projective* order, since it is attained by projecting the scoring rule onto individual coordinates. Given a query $q$, we evaluate web pages in the lists $L_i$ that correspond to features of $q$. The algorithm maintains two statistics, upper and lower bounds on the score of the top-$k^{th}$ page, halting when these bounds cross. The lower bound is the score of the $k^{th}$ best page seen so far; the upper bound is the sum of the partial scores (i.e., $g_i(p)$) for the next-to-be-scored page in each list. Since the lists are ordered by the partial scores, the upper threshold does in fact bound the score of any page yet to be seen.

The threshold algorithm is particularly effective when a query contains a small number of features, facilitating fast convergence of the upper bound. In our experiments, we find that the halting condition is rarely satisfied within the imposed computational restrictions. One can, of course, simply halt the algorithm when it has expended the computational budget (Fagin, 2002), which we refer to as the *Halted Threshold Algorithm*.

**Inverted Indices.** An inverted index is a datastructure that maps every page feature $x$ to a list of pages $p$ that contain $x$. When a new query arrives, a subset of page features relevant to the query is first determined. For instance, when the query contains "dog", the page feature set might be {"dog", "canine", "collar", ...}. Note that a distinction is made between query features and page features, and in particular, the relevant page features may include many more words than the query itself. Once a set of page features is determined, their respective lists (i.e., inverted indices) are searched, and from them the final list of output pages is chosen. One method for searching over these lists is to execute Fagin's threshold algorithm. Other methods, such as the "Weighted-And" algorithm (Broder et al., 2003), use one global order for pages in the lists and walk down the lists synchronously to compute page scores. See (Zobel & Moffat, 2006) for an overview of inverted indices applied to web search.

Standard approaches based on inverted indices suffer from a shortcoming. The resulting algorithms are efficient only when it is sufficient to search over a relatively small set of inverted indices for each

query. They require, for each query $q$, that there exists a small set[2] $X_q$ of page features such that the score of any page against $q$ depends only on its intersection with $X_q$. In other words, the scoring rule must be extremely sparse, with most words or features in the page having zero contribution to the score for $q$. In Section 3.1, we consider a machine-learned scoring rule, derived from internet advertising data, with the property that almost all page features have substantial influence on the score for every query, making any straightforward approach based on inverted indices intractable. Furthermore, algorithms that use inverted indices do not typically optimize the datastructure against the query distribution and our experiments suggest that doing so may be beneficial.

## 2 An Algorithm for Rapid Approximate Ranking

Suppose we are provided with a categorization of possible queries into related, potentially overlapping, sets. For example, these sets might be defined as, "queries containing the word 'France'," or "queries with the phrase 'car rental'." For each query set, the associated predictive index is an ordered list of web pages sorted by their *expected* score for random queries drawn from that set. In particular, we expect web pages at the top of the 'France' list to be good, on average, for queries containing the word 'France.' In contrast to an inverted index, the pages in the 'France' list need not themselves contain the word 'France'. To retrieve results for a particular query (e.g., "France car rental"), we optimize only over web pages in the relevant, pre-computed lists. Note that the predictive index is built on top of an already existing categorization of queries, a critical, and potentially difficult initial step. In the applications we consider, however, we find that predictive indexing works well even when applied to naively defined query sets. Furthermore, in our application to approximate nearest neighbor search, we found predictive indexing to be robust to cover sets generated via random projections whose size and shape were varied across experiments.

We represent queries and web pages as points in, respectively, $Q \subseteq \mathbb{R}^n$ and $W \subseteq \mathbb{R}^m$. This setting is general, but for the experimental application we consider $n, m \approx 10^6$, with any given page or query having about $10^2$ non-zero entries (see Section 3.1 for details). Thus, pages and points are typically sparse vectors in very high dimensional spaces. A coordinate may indicate, for example, whether a particular word is present in the page/query, or more generally, the number of times that word appears. Given a scoring function $f : Q \times W \mapsto \mathbb{R}$, and a query $q$, we attempt to rapidly find the top-$k$ pages $p_1, \ldots, p_k$. Typically, we find an approximate solution, a set of pages $\hat{p}_1, \ldots, \hat{p}_k$ that are among the top $l$ for $l \approx k$. We assume queries are generated from a probability distribution $D$ that may be sampled.

### 2.1 Predictive Indexing for General Scoring Functions

Consider a finite collection $\mathcal{Q}$ of sets $Q_i \subseteq Q$ that cover the query space (i.e., $Q \subseteq \cup_i Q_i$). For each $Q_i$, define the conditional probability distribution $D_i$ over queries in $Q_i$ by $D_i(\cdot) = D(\cdot|Q_i)$, and define $f_i : W \mapsto \mathbb{R}$ as $f_i(p) = \mathbb{E}_{q \sim D_i}[f(q, p)]$. The function $f_i(p)$ is the expected score of the web page $p$ for the (related) queries in $Q_i$. The hope is that any page $p$ has approximately the same score for any query $q \in Q_i$. If, for example, $Q_i$ is the set of queries that contain the word "dog", we may expect every query in $Q_i$ to score high against pages about dogs and to score low against those pages not about dogs.

For each set of queries $Q_i$ we pre-compute a sorted list $L_i$ of pages $p_{i_1}, p_{i_2}, \ldots, p_{i_N}$ ordered in descending order of $f_i(p)$. At runtime, given a query $q$, we identify the query sets $Q_i$ containing $q$, and compute the scoring function $f$ only on the restricted set of pages at the beginning of their associated lists $L_i$. We search down these lists for as long as the computational budget allows.

In general, it is difficult to compute exactly the conditional expected scores of pages $f_i(p)$. One can, however, approximate these scores by sampling from the query distribution $D$. Algorithm 1 outlines the construction of the sampling-based predictive indexing datastructure. Algorithm 2 shows how the method operates at run time.

Note that in the special case where we cover $Q$ with a single set, we end up with a global ordering of web pages, independent of the query, which is optimized for the underlying query distribution.

**Algorithm 1** Construct-Predictive-Index(Cover $\mathcal{Q}$, Dataset $S$)

---

    $L_j[s] = 0$ for all objects $s$ and query sets $Q_j$.
    **for** $t$ random queries $q \sim D$ **do**
        **for all** objects $s$ in the data set **do**
            **for all** query sets $Q_j$ containing $q$ **do**
                $L_j[s] \leftarrow L_j[s] + f(q, s)$
            **end for**
        **end for**
    **end for**

    **for all** lists $L_j$ **do**
        sort $L_j$
    **end for**

    **return** $\{L\}$

---

**Algorithm 2** Find-Top(query $q$, count $k$)

---

    $i = 0$
    top-$k$ list $V = \emptyset$
    **while** time remains **do**
        **for** each query set $Q_j$ containing $q$ **do**
            $s \leftarrow L_j[i]$
            **if** $f(q, s) > k^{th}$ best seen so far **then**
                insert $s$ into ordered top-$k$ list $V$
            **end if**
        **end for**
        $i \leftarrow i + 1$
    **end while**

    **return** $V$

---

While this global ordering may not be effective in isolation, it could perhaps be used to order pages in traditional inverted indices.

## 2.2 Discussion

We present an elementary example to help develop intuition for why we can sometimes expect predictive indexing to improve upon projective datastructures such as those used in Fagin's threshold algorithm. Suppose we have: two query features $t_1$ and $t_2$; three possible queries $q_1 = \{t_1\}$, $q_2 = \{t_2\}$ and $q_3 = \{t_1, t_2\}$; and three web pages $p_1$, $p_2$ and $p_3$. Further suppose we have a simple linear scoring function defined by

$$f(q, p_1) = I_{t_1 \in q} - I_{t_2 \in q} \qquad f(q, p_2) = I_{t_2 \in q} - I_{t_1 \in q} \qquad f(q, p_3) = .5 \cdot I_{t_2 \in q} + .5 \cdot I_{t_1 \in q}$$

where $I$ is the indicator function. That is, $p_i$ is the best match for query $q_i$, but $p_3$ does not score highly for either query feature alone. Thus, an ordered, projective datastructure would have

$$t_1 \rightarrow \{p_1, p_3, p_2\} \qquad t_2 \rightarrow \{p_2, p_3, p_1\}.$$

Suppose, however, that we typically only see query $q_3$. In this case, if we know $t_1$ is in the query, we infer that $t_2$ is likely to be in the query (and vice versa), and construct the predictive index

$$t_1 \rightarrow \{p_3, p_1, p_2\} \qquad t_2 \rightarrow \{p_3, p_2, p_1\}.$$

On the high probability event, namely query $q_3$, we see the predictive index outperforms the projective, query independent, index.

We expect predictive indices to generally improve on datastructures that are agnostic to the query distribution. In the simple case of a single cover set (i.e., a global web page ordering) and when we wish to optimize the probability of returning the highest-scoring object, Lemma 2.1 shows that a predictive ordering is the best ordering relative to any particular query distribution.

**Lemma 2.1.** *Suppose we have a set of points $S$, a query distribution $D$, and a function $f$ that scores queries against points in $S$. Further assume that for each query $q$, there is a unique highest scoring point $H_q$. For $s \in S$, let $h(s) = \Pr_{q \sim D}(s = H_q)$, and let $s_1, s_2, \ldots, s_N$ be ordered according to $h(s)$. For any fixed $k$,*

$$\Pr_{q \sim D}\left(H_q \in \{s_1, ..., s_k\}\right) = \max_{\text{permutations } \pi} \Pr_{q \sim D}\left(H_q \in \{s_{\pi(1)}, ..., s_{\pi(k)}\}\right).$$

*Proof.* For any ordering of points, $s_{\pi(1)}, \ldots, s_{\pi(k)}$, the probability of the highest scoring point appearing in the top $k$ entries equals $\sum_{j=1}^{k} h(s_{\pi(j)})$. This sum is clearly maximized by ordering the list according to $h(\cdot)$. $\square$

## 3 Empirical Evaluation

We evaluate predictive indexing for two applications: Internet advertising and approximate nearest neighbor.

### 3.1 Internet Advertising

We present results on *Internet advertising*, a problem closely related to web search. We have obtained proprietary data, both testing and training, from an online advertising company. The data are comprised of logs of events, where each event represents a visit by a user to a particular web page $p$, from a set of web pages $Q \subseteq \mathbb{R}^n$. From a large set of advertisements $W \subseteq \mathbb{R}^m$, the commercial system chooses a smaller, ordered set of ads to display on the page (generally around 4). The set of ads seen and clicked by users is logged. Note that the role played by web pages has switched, from result to query. The total number of ads in the data set is $|W| \approx 6.5 \times 10^5$. Each ad contains, on average, 30 ad features, and a total of $m \approx 10^6$ ad features are observed. The training data consist of 5 million events (web page $\times$ ad displays). The total number of distinct web pages is $5 \times 10^5$. Each page consists of approximately 50 page features, and a total of $n \approx 9 \times 10^5$ total page features are observed.

We used a sparse feature representation (see Section 1.1) and trained a linear scoring rule $f$ of the form $f(p, a) = \sum_{i,j} w_{i,j} p_i a_j$, to approximately rank the ads by their probability of click. Here, $w_{i,j}$ are the learned weights (parameters) of the linear model. The search algorithms we compare were given the scoring rule $f$, the training pages, and the ads $W$ for the necessary pre-computations. They were then evaluated by their serving of $k = 10$ ads, under a time constraint, for each page in the test set. There was a clear separation of test and training. We measured computation time in terms of the number of full evaluations by the algorithm (i.e., the number of ads scored against a given page). Thus, the true test of an algorithm was to quickly select the most promising $T$ ads to fully score against the page, where $T \in \{100, 200, 300, 400, 500\}$ was externally imposed and varied over the experiments. These numbers were chosen to be in line with real-world computational constraints.

We tested four methods: **halted threshold algorithm** (TA), as described in Section 1.2, two variants of predictive indexing (**PI-AVG** and **PI-DCG**), and a fourth method, called **best global ordering** (BO), which is a degenerate form of PI discussed in Section 2.1. An inverted index approach is prohibitively expensive since almost all ad features have substantial influence on the score for every web page (see Section 1.2).

PI-AVG and PI-DCG require a covering of the web page space. We used the natural covering suggested by the binary features—each page feature $i$ corresponds to a cover set consisting of precisely those pages $p$ that contain $i$. The resulting datastructure is therefore similar to that maintained by the TA algorithm—lists, for each page feature, containing all the ads. However, while TA orders ads by partial score $\sum_j w_{i,j} p_i a_j$ for each fixed page feature $i$, the predictive methods order by *expected* score. PI-AVG sorts ad lists by expected score of $f$, $\mathbb{E}_{p \sim D_i}[f(p, a)] = \mathbb{E}_{p \sim D}[f(p, a) | i \in p]$, conditioned on the page containing feature $i$. PI-DCG and BO optimize the expected value of a modified scoring rule, $\mathrm{DCG}_f(p, a) = I_{r(p,a) \leq 16} / \log_2(r(p, a) + 1)$, where $r$ is the rank function and $I$ is the indicator function. Here, $r(p, a) = j$ indicates that ad $a$ has rank $j$ according to $f(p, a)$ over all ads in $W$. BO stores a single list of all ads, sorted by expected $\mathrm{DCG}_f(p, a)$, while PI-DCG stores a list for each page feature $i$ sorted by $\mathbb{E}_{p \sim D_i}[\mathrm{DCG}_f(p, a)]$. We chose this measure because:

1. Compared with using the average score of $f$, we empirically observe that expected $\mathrm{DCG}_f$ greatly improves the performance of BO on these data.

2. It is related to "discounted cumulative gain", a common measure for evaluating search results in the information retrieval literature (Järvelin & Kekäläinen, 2002).

3. Expected $\mathrm{DCG}_f$ is zero for many ads, enabling significant compression of the predictive index.

4. Lemma 2.1 suggests ordering by the probability an ad is in the top 10. The $\mathrm{DCG}_f$ score is a softer version of indicator of top 10.

All three predictive methods were trained by sampling from the training set, as described in 2.1.

Figure 3.1 plots the results of testing the four algorithms on the web advertising data. Each point in the figure corresponds to one experiment, which consisted of executing each algorithm on $1000$ test pages. Along the $x$-axis we vary the time constraint imposed on the algorithm. The $y$-axis plots the frequency, over the test pages, that the algorithm succeeded in serving the top scoring ad for position 1 (Figure 1(a)) and for position 10 (Figure 1(b)). Thus, vertical slices through each plot show the difference in performance between the algorithms when they are given the same amount of serving time per page. The probabilities were computed by off-line scoring of all $6.5 \times 10^5$ ads for each test page and computing the true top-10 ads. Serving correctly for position 10 is more difficult than for position 1, because it also requires correctly serving ads for positions 1 through 9. We see that all three methods of predictive indexing are superior to Fagin's halted threshold algorithm. In addition, the use of a richer covering, for PI-DCG and PI-AVG, provides a large boost in performance. These latter two predictive indexing methods attain relatively high accuracy even when fully evaluating only 0.05% of the potential results.

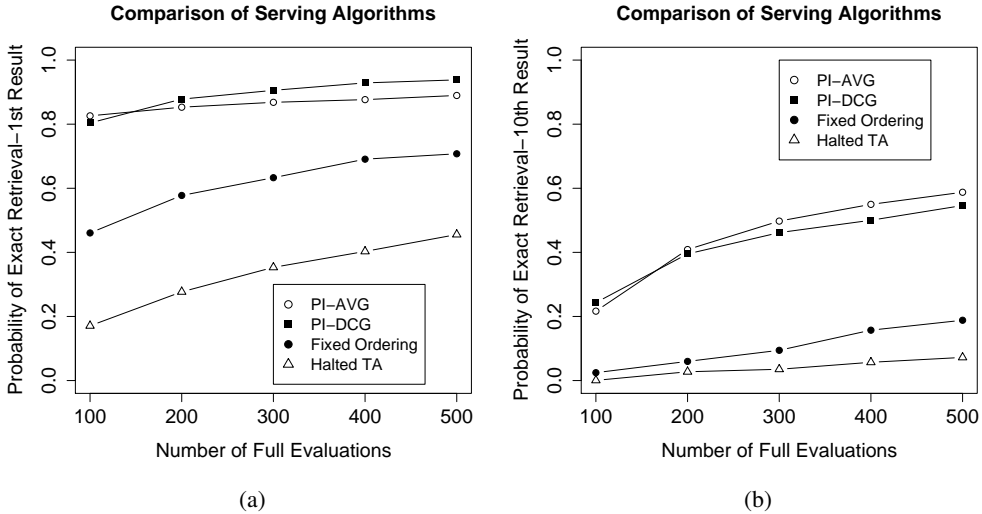

Figure 1: Comparison of the first and tenth results returned from the four serving algorithms on the web advertisement dataset.

Our implementation of the predictive index, and also the halted threshold algorithm, required about 50ms per display event when 500 ad evaluations are allowed. The RAM use for the predictive index is also reasonable, requiring about a factor of 2 more RAM than the ads themselves.

## 3.2 Approximate Nearest Neighbor Search

A special case application of predictive indexing is *approximate nearest neighbor search*. Given a set of points $W$ in $n$-dimensional Euclidean space, and a query point $x$ in that same space, the nearest neighbor problem is to quickly return the top-$k$ neighbors of $x$. This problem is of considerable interest for a variety of applications, including data compression, information retrieval, and pattern recognition. In the predictive indexing framework, the nearest neighbor problem corresponds to

minimizing a scoring function, $f(x, y) = ||x - y||_2$, defined by Euclidean distance. We assume query points are generated from a distribution $D$ that can be sampled.

To start, we define a covering $\mathcal{Q}$ of the input space $\mathbb{R}^n$, which we borrow from locality-sensitive hashing (LSH) (Gionis et al., 1999; Datar et al., 2004), a commonly suggested scheme for the approximate nearest neighbor problem. Fix positive integer parameters $\alpha, \beta$. First, we form $\alpha$ random partitions of the input space. Geometrically, each partition splits the $n$-dimensional space on $\beta$ random hyperplanes. Formally, for all $1 \le i \le \alpha$ and $1 \le j \le \beta$, generate a random unit-norm $n$-vector $Y^{ij} = (Y_1^{ij}, \ldots, Y_n^{ij}) \in \mathbb{R}^n$ from the Gaussian (normal) distribution. For fixed $i \in \{1, \ldots, \alpha\}$ and subset $J \subseteq \{1, \ldots, \beta\}$ define the cover set $Q_{i,J} = \{x \in \mathbb{R}^n : x \cdot Y^{ij} \ge 0 \text{ if and only if } j \in J\}$. Note that for fixed $i$, the set $\{Q_{i,J} | J \subseteq \{1, \ldots, k\}\}$ partitions the space by random planes.

Given a query point $x$, consider the union $U_x = \bigcup_{\{Q_{i,J} \in \mathcal{Q} \mid x \in Q_{i,J}\}} Q_{i,J}$ of all cover sets containing $x$. Standard LSH approaches to the nearest neighbor problem work by scoring points in the set $Q_x = W \cap U_x$. That is, LSH considers only those points in $W$ that are covered by at least one of the same $\alpha$ sets as $x$. Predictive indexing, in contrast, maps each cover set $Q_{i,J}$ to an ordered list of points sorted by their *probability of being a top-10 nearest point* to points in $Q_{i,J}$. That is, the lists are sorted by $h_{Q_{i,J}}(p) = \Pr_{q \sim D|Q_{i,J}}(p \text{ is one of the nearest 10 points to } q)$. For the query $x$, we then consider those points in $W$ with large probability $h_{Q_{i,J}}$ for at least one of the sets $Q_{i,J}$ that cover $x$.

We compare LSH and predictive indexing over four data sets: (1) MNIST—60,000 training and 10,000 test points in 784 dimensions; (2) Corel—37,749 points in 32 dimensions, split randomly into 95% training and 5% test subsets; (3) Pendigits—7494 training and 3498 test points in 17 dimensions; and (4) Optdigits—3823 training and 1797 test points in 65 dimensions. The MNIST data is available at `http://yann.lecun.com/exdb/mnist/` and the remaining three data sets are available at the UCI Machine Learning Repository (`http://archive.ics.uci.edu/ml/`). Random projections were generated for each experiment, inducing a covering of the space that was provided to both LSH and predictive indexing. The predictive index was generated by sampling over the training set as discussed in Section 2.1. The number of projections $\beta$ per partition was set to 24 for the larger sets (Corel and MNIST) and 63 for the smaller sets (Pendigits and Optdigits), while the number of partitions $\alpha$ was varied as an experimental parameter. Larger $\alpha$ corresponds to more full evaluations per query, resulting in improved accuracy at the expense of increased computation time. Both algorithms were restricted to the same average number of full evaluations per query.

Predictive indexing offers substantial improvements over LSH for all four data sets. Figure 2(a) displays the true rank of the first point returned by LSH and predictive indexing on the MNIST data set as a function of $\alpha$, averaged over all points in the test set and over multiple trials. Predictive indexing outperforms LSH at each parameter setting, with the difference particularly noticeable when fewer full evaluation are permitted (i.e., small $\alpha$). Figure 2(b) displays the performance of LSH and predictive indexing for the tenth point returned, over all four data sets, with values of $\alpha$ varying from 5 to 70, averaged over the test sets, and replicated by multiple runs. In over 300 trials, we did not observe a single instance of LSH outperforming predictive indexing.

Recent work has proposed more sophisticated partitionings for LSH (Andoni & Indyk, 2006). Approaches based on metric trees (Liu et al., 2004), which take advantage of the distance metric structure, have also been shown to perform well for approximate nearest neighbor. Presumably, taking advantage of the query distribution could further improve these algorithms as well, although that is not studied here.

# 4   Conclusion

Predictive indexing is the first datastructure capable of supporting scalable, rapid ranking based on general purpose machine-learned scoring rules. In contrast, existing alternatives such as the Threshold Algorithm (Fagin, 2002) and Inverted Index approaches (Broder et al., 2003) are either substantially slower, inadequately expressive, or both, for common machine-learned scoring rules. In the special case of approximate nearest neighbors, predictive indexing offers substantial and consistent improvements over the Locality Sensitive Hashing algorithm.

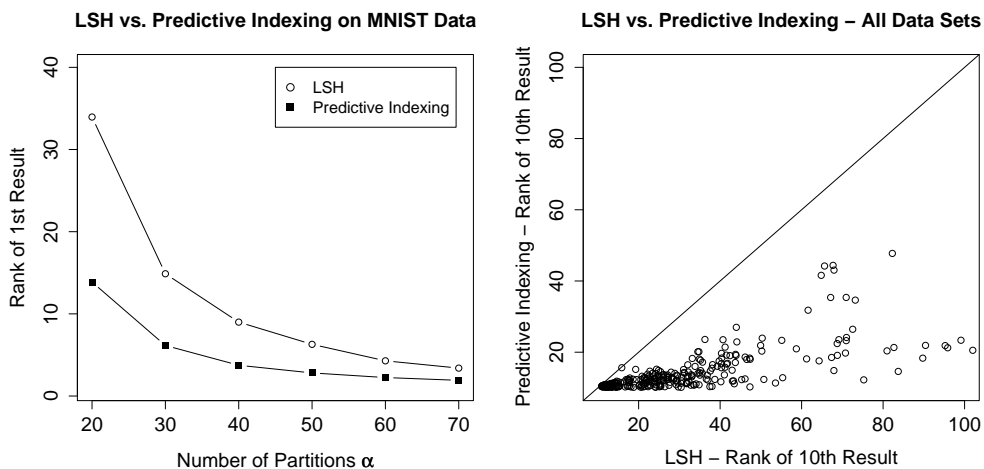

(a) The y-axis, "Rank of 1st Result" measures the true rank of the first result returned by each method. As the number of partitions $\alpha$ is increased, improved accuracy is achieved at the expense of longer computation time.

(b) Each point represents the outcome of a single experiment for one of the four data sets at various parameter settings.

Figure 2: Comparison of the first and tenth results returned from LSH and predictive indexing.

## Footnotes

[1]More general *monotone* scoring functions (e.g., coordinate-wise product and $\max$) are in fact supported; for clarity, however, we restrict to the linear case.

[2]The size of these sets are typically on the order of 100 or smaller.

# References

Andoni, A., & Indyk, P. (2006). Near-optimal hashing algorithms for near neighbor problem in high dimensions. *Proceedings of the Symposium on Foundations of Computer Science (FOCS'06)*.

Broder, A. Z., Carmel, D., Herscovici, M., Soffer, A., & Zien, J. (2003). Efficient query evaluation using a two-level retrieval process. *CIKM '03: Proceedings of the twelfth international conference on Information and knowledge management* (pp. 426–434).

Cheng, C.-S., Chung, C.-P., & Shann, J. J.-J. (2006). Fast query evaluation through document identifier assignment for inverted file-based information retrieval systems. *Inf. Process. Manage.*, *42*, 729–750.

Chierichetti, F., Lattanzi, S., Mari, F., & Panconesi, A. (2008). On placing skips optimally in expectation. *WSDM '08: Proceedings of the international conference on Web search and web data mining* (pp. 15–24). New York, NY, USA: ACM.

Datar, M., Immorlica, N., Indyk, P., & Mirrokni, V. S. (2004). Locality-sensitive hashing scheme based on p-stable distributions. *SCG '04: Proceedings of the twentieth annual symposium on Computational geometry* (pp. 253–262). New York, NY, USA: ACM.

Fagin, R. (2002). Combining fuzzy information: an overview. *SIGMOD Rec.*, *31*, 109–118.

Fagin, R., Lotem, A., & Naor, M. (2003). Optimal aggregation algorithms for middleware. *J. Comput. Syst. Sci.*, *66*, 614–656.

Gionis, A., Indyk, P., & Motwani, R. (1999). Similarity search in high dimensions via hashing. *The VLDB Journal* (pp. 518–529).

Järvelin, K., & Kekäläinen, J. (2002). Cumulated gain-based evaluation of IR techniques. *ACM Transactions on Information Systems*, *20*, 422–446.

Liu, T., Moore, A., Gray, A., & Yang, K. (2004). An investigation of practical approximate nearest neighbor algorithms. *Neural Information Processing Systems*.

Zobel, J., & Moffat, A. (2006). Inverted files for text search engines. *ACM Comput. Surv.*, *38*, 6.

